# Efficient Convex Relaxation for Transductive Support Vector Machine

**Zenglin Xu**
Dept. of Computer Science & Engineering
The Chinese University of Hong Kong
Shatin, N.T., Hong Kong
zlxu@cse.cuhk.edu.hk

**Rong Jin**
Dept. of Computer Science & Engineering
Michigan State University
East Lansing, MI, 48824
rongjin@cse.msu.edu

**Jianke Zhu**    **Irwin King**    **Michael R. Lyu**
Dept. of Computer Science & Engineering
The Chinese University of Hong Kong
Shatin, N.T., Hong Kong
{jkzhu,king,lyu}@cse.cuhk.edu.hk

## Abstract

We consider the problem of Support Vector Machine transduction, which involves a combinatorial problem with exponential computational complexity in the number of unlabeled examples. Although several studies are devoted to Transductive SVM, they suffer either from the high computation complexity or from the solutions of local optimum. To address this problem, we propose solving Transductive SVM via a convex relaxation, which converts the NP-hard problem to a semi-definite programming. Compared with the other SDP relaxation for Transductive SVM, the proposed algorithm is computationally more efficient with the number of free parameters reduced from $\mathcal{O}(n^2)$ to $\mathcal{O}(n)$ where $n$ is the number of examples. Empirical study with several benchmark data sets shows the promising performance of the proposed algorithm in comparison with other state-of-the-art implementations of Transductive SVM.

## 1   Introduction

Semi-supervised learning has attracted an increasing amount of research interest recently [3, 15]. An important semi-supervised learning paradigm is the Transductive Support Vector Machine (TSVM), which maximizes the margin in the presence of unlabeled data and keeps the boundary traversing through low density regions, while respecting labels in the input space.

Since TSVM requires solving a combinatorial optimization problem, extensive research efforts have been devoted to efficiently finding the approximate solution to TSVM. The popular version of TSVM proposed in [8] uses a label-switching-retraining procedure to speed up the computation. In [5], the hinge loss in TSVM is replaced by a smooth loss function, and a gradient descent method is used to find the decision boundary in a region of low density. Chapelle et al. [2] employ an iterative approach for TSVM. It begins with minimizing an easy convex object function, and then gradually approximates the objective of TSVM with more complicated functions. The solution of the simple function is used as the initialization for the solution to the complicated function. Other iterative methods, such as deterministic annealing [11] and the concave-convex procedure (CCCP) method [6], are also employed to solve the optimization problem related to TSVM. The main drawback of the approximation methods listed above is that they are susceptible to local optima, and therefore are sensitive to the initialization of solutions. To address this problem, in [4], a branch-

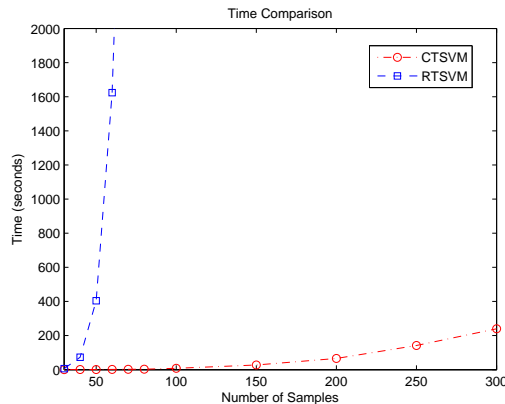

Figure 1: Computation time of the proposed convex relaxation approach for TSVM (i.e., CTSVM) and the semi-definite relaxation approach for TSVM (i.e., RTSVM) versus the number of unlabeled examples. The Course data set is used, and the number of labeled examples is 20.

and-bound search method is developed to find the exact solution. In [14], the authors approximate TSVM by a semi-definite programming problem, which leads to a relaxation solution to TSVM (noted as RTSVM), to avoid the solution of local optimum. However, both approaches suffer from the high computational cost and can only be applied to small sized data sets.

To this end, we present the convex relaxation for Transductive SVM (**CTSVM**). The key idea of our method is to approximate the non-convex optimization problem of TSVM by its dual problem. The advantage of doing so is twofold:

- Unlike the semi-definite relaxation [14] that approximates TSVM by dropping the rank constraint, the proposed approach approximates TSVM by its dual problem. As the basic result of convex analysis, the conjugate of conjugate of any function $f(\mathbf{x})$ is the convex envelope of $f(\mathbf{x})$, and therefore provides a tighter convex relaxation for $f(\mathbf{x})$ [7]. Hence, the proposed approach provides a better convex relaxation than that in [14] for the optimization problem in TSVM.

- Compared to the semi-definite relaxation TSVM, the proposed algorithm involves fewer free parameters and therefore significantly improves the efficiency by reducing the worst-case computational complexity from $\mathcal{O}(n^{6.5})$ to $\mathcal{O}(n^{4.5})$. Figure 1 shows the running time of both the semi-definite relaxation of TSVM in [14] and the proposed convex relaxation for TSVM versus increasing number of unlabeled examples. The data set used in this example is the Course data set (see the experiment section), and the number of labeled examples is 20. We clearly see that the proposed convex relaxation approach is considerably more efficient than the semi-definition approach.

The rest of this paper is organized as follows. Section 2 reviews the related work on the semi-definite relaxation for TSVM. Section 3 presents the convex relaxation approach for Transductive SVM. Section 4 presents the empirical studies that verify the effectiveness of the proposed relaxation for TSVM. Section 5 sets out the conclusion.

## 2 Related Work

In this section, we review the key formulae for Transductive SVM, followed by the semi-definite programming relaxation for TSVM.

Let $\mathcal{X} = (\mathbf{x}_1, \ldots, \mathbf{x}_n)$ denote the entire data set, including both the labeled examples and the unlabeled ones. We assume that the first $l$ examples within $\mathcal{X}$ are labeled by $\mathbf{y}_\ell = (y_1^\ell, y_2^\ell, \ldots, y_l^\ell)$ where $y_i^\ell \in \{-1, +1\}$ represents the binary class label assigned to $\mathbf{x}_i$. We further denote by $\mathbf{y} = (y_1, y_2, \ldots, y_n) \in \{-1, +1\}^n$ the binary class labels predicted for all the data points in $\mathcal{X}$. The goal of TSVM is to estimate $\mathbf{y}$ by using both the labeled examples and the unlabeled ones.

Following the framework of maximum margin, TSVM aims to identify the classification model that will result in the maximum classification margin for both labeled and unlabeled examples, which amounts to solve the following optimization problem:

$$\min_{\mathbf{w},b,\mathbf{y}\in\{-1,+1\}^n,\varepsilon} \quad \|\mathbf{w}\|_2^2 + C\sum_{i=1}^n \varepsilon_i$$

$$\text{s. t.} \quad y_i(\mathbf{w}^\top \mathbf{x}_i - b) \geq 1 - \varepsilon_i, \varepsilon_i \geq 0, \ i = 1, 2, \ldots, n$$

$$y_i = y_i^\ell, \ i = 1, 2, \ldots, l,$$

where $C \geq 0$ is the trade-off parameter between the complexity of function $\mathbf{w}$ and the margin errors. The prediction function can be formulated as $f(\mathbf{x}) = \mathbf{w}^\top \mathbf{x} - b$.

Evidently, the above problem is a non-convex optimization problem due to the product term $y_i w_j$ in the constraint. In order to approximate the above problem into a convex programming problem, we first rewrite the above problem into the following form using the Lagrange Theorem:

$$\min_{\nu,\mathbf{y}\in\{-1,+1\}^n,\delta,\lambda} \quad \frac{1}{2}(\mathbf{e}+\nu-\delta+\lambda\mathbf{y})^\top \mathcal{D}(\mathbf{y})\mathbf{K}^{-1}\mathcal{D}(\mathbf{y})(\mathbf{e}+\nu-\delta+\lambda\mathbf{y}) + C\delta^\top \mathbf{e} \quad (1)$$

$$\text{s. t.} \quad \nu \geq 0, \quad \delta \geq 0, \quad y_i = y_i^\ell, \ i = 1, 2, \ldots, l,$$

where $\nu$, $\delta$ and $\lambda$ are the dual variables. $\mathbf{e}$ is the $n$-dimensional column vector of all ones and $\mathbf{K}$ is the kernel matrix. $\mathcal{D}(\mathbf{y})$ represents a diagonal matrix whose diagonal elements form the vector $\mathbf{y}$. Detailed derivation can be found in [9, 13]. Using the Schur complement, the above formulation can be further formulated as follows:

$$\min_{\mathbf{y}\in\{-1,+1\}^n,t,\nu,\delta,\lambda} \quad t \qquad\qquad\qquad\qquad\qquad\qquad (2)$$

$$\text{s. t.} \quad \begin{pmatrix} \mathbf{y}\mathbf{y}^\top \circ \mathbf{K} & \mathbf{e}+\nu-\delta+\lambda\mathbf{y} \\ (\mathbf{e}+\nu-\delta+\lambda\mathbf{y})^\top & t - 2C\delta^\top \mathbf{e} \end{pmatrix} \succeq 0$$

$$\nu \geq 0, \ \delta \geq 0, \ y_i = y_i^\ell, \ i = 1, 2, \ldots, l,$$

where the operator $\circ$ represents the element wise product.

To convert the above problem into a convex optimization problem, the key idea is to replace the quadratic term $\mathbf{y}\mathbf{y}^\top$ by a linear variable. Based on the result that the set $\mathcal{S}_a = \{\mathbf{M} = \mathbf{y}\mathbf{y}^\top | \mathbf{y} \in \{-1,+1\}^n\}$ is equivalent to the set $\mathcal{S}_b = \{\mathbf{M} | M_{i,i} = 1, \text{rank}(\mathbf{M}) = 1\}$, we can approximate the problem in (2) as follows:

$$\min_{\mathbf{M},t,\nu,\delta,\lambda} \quad t \qquad\qquad\qquad\qquad\qquad\qquad (3)$$

$$\text{s. t.} \quad \begin{pmatrix} \mathbf{M} \circ \mathbf{K} & \mathbf{e}+\nu-\delta \\ (\mathbf{e}+\nu-\delta)^\top & t - 2C\delta^\top \mathbf{e} \end{pmatrix} \succeq 0$$

$$\nu \geq 0, \ \delta \geq 0,$$

$$\mathbf{M} \succeq 0, \ M_{i,i} = 1, \ i = 1, 2, \ldots, n,$$

where $M_{ij} = y_i^\ell y_j^\ell$ for $1 \leq i, j \leq l$.

Note that the key differences between (2) and (3) are (a) the rank constraint $\text{rank}(\mathbf{M}) = 1$ is removed, and (b) the variable $\lambda$ is set to be zero, which is equivalent to setting $b = 0$. The above approximation is often referred to as the Semi-Definite Programming (SDP) relaxation. As revealed by the previous studies [14, 1], the SDP programming problem resulting from the approximation is computationally expensive. More specifically, there are $\mathcal{O}(n^2)$ parameters in the SDP cone and $\mathcal{O}(n)$ linear inequality constraints, which implies a worst-case computational complexity of $\mathcal{O}(n^{6.5})$. To avoid the high computational complexity, we present a different approach for relaxing TSVM into a convex problem. Compared to the SDP relaxation approach, it is advantageous in that (1) it produces the best convex approximation for TSVM, and (2) it is computationally more efficient than the previous SDP relaxation.

## 3 Relaxed Transductive Support Vector Machine

In this section, we follow the work of generalized maximum margin clustering [13] by first studying the case of hard margin, and then extending it to the case of soft margin.

### 3.1 Hard Margin TSVM

In the hard margin case, SVM does not penalize the classification error, which corresponds to $\delta = 0$ in (1). The resulting formulism of TSVM becomes

$$\min_{\nu, \mathbf{y}, \lambda} \quad \frac{1}{2}(\mathbf{e} + \nu + \lambda\mathbf{y})^\top \mathcal{D}(\mathbf{y})\mathbf{K}^{-1}\mathcal{D}(\mathbf{y})(\mathbf{e} + \nu + \lambda\mathbf{y}) \tag{4}$$

$$s.\,t. \quad \nu \geq 0,$$
$$y_i = y_i^\ell, \; i = 1, 2, \ldots, l,$$
$$y_i^2 = 1, \; i = l+1, l+2, \ldots, n.$$

Instead of employing the SDP relaxation as in [14], we follow the work in [13] and introduce a variable $\mathbf{z} = \mathcal{D}(\mathbf{y})(\mathbf{e} + \nu) = \mathbf{y} \circ (\mathbf{e} + \nu)$. Given that $\nu \geq 0$, the constraints in (4) can be written as $y_i^\ell z_i \geq 1$ for the labeled examples, and $z_i^2 \geq 1$ for all the unlabeled examples. Hence, $\mathbf{z}$ can be used as the prediction function, i.e., $f^* = \mathbf{z}$. Using this new notation, the optimization problem in (4) can be rewritten as follows:

$$\min_{\mathbf{z}, \lambda} \quad \frac{1}{2}(\mathbf{z} + \lambda\mathbf{e})^\top \mathbf{K}^{-1}(\mathbf{z} + \lambda\mathbf{e}) \tag{5}$$

$$s.\,t. \quad y_i^\ell z_i \geq 1, \; i = 1, 2, \ldots, l,$$
$$z_i^2 \geq 1, \; i = l+1, l+2, \ldots, n.$$

One problem with Transductive SVMs is that it is possible to classify all the unlabeled data to one of the classes with a very large margin due to the high dimension and few labeled data. This will lead to poor generalization ability. To solve this problem, we introduce the following balance constraint to ensure that no class takes all the unlabeled examples:

$$-\epsilon \leq \frac{1}{l}\sum_{i=1}^{l} z_i - \frac{1}{n-l}\sum_{i=l+1}^{n} z_i \leq \epsilon, \tag{6}$$

where $\epsilon \geq 0$ is a constant. Through the above constraint, we aim to ensure that the difference between the labeled data and the unlabeled data in their class assignment is small.

To simplify the expression, we further define $\mathbf{w} = (\mathbf{z}, \lambda) \in \mathbb{R}^{n+1}$ and $\mathbf{P} = (\mathbf{I}_n, \mathbf{e}) \in \mathbb{R}^{n \times (n+1)}$. Then, the problem in (5) becomes:

$$\min_{\mathbf{w}} \quad \mathbf{w}^\top \mathbf{P}^\top \mathbf{K}^{-1} \mathbf{P} \mathbf{w} \tag{7}$$

$$s.\,t. \quad y_i^\ell w_i \geq 1, \; i = 1, 2, \ldots, l,$$
$$w_i^2 \geq 1, \; i = l+1, l+2, \ldots, n,$$
$$-\epsilon \leq \frac{1}{l}\sum_{i=1}^{l} w_i - \frac{1}{n-l}\sum_{i=l+1}^{n} w_i \leq \epsilon.$$

When this problem is solved, the label vector $\mathbf{y}$ can be directly determined by the sign of the prediction function, i.e., $\text{sign}(\mathbf{w})$. This is because $w_i = (1 + \nu)y_i$ for $i = l+1, \ldots, n$ and $\nu \geq 0$.

The following theorem shows that the problem in (7) can be relaxed to a semi-definite programming.

**Theorem 1.** Given a sample $\mathcal{X} = \{\mathbf{x}_1, \ldots, \mathbf{x}_n\}$ and a partial set of the labels $\mathbf{y}_\ell = (y_1^\ell, y_2^\ell, \ldots, y_l^\ell)$ where $1 \leq l \leq n$, the variable $\mathbf{w}$ that optimizes (7) can be calculated by

$$\mathbf{w} = \frac{1}{2}\left[\mathbf{A} - \mathcal{D}(\gamma \circ \mathbf{b})\right]^{-1}(\gamma \circ \mathbf{a} - (\alpha - \beta)\mathbf{c}), \tag{8}$$

where $\mathbf{a} = (\mathbf{y}^l, \mathbf{0}^{n-l}, 0) \in \mathbb{R}^{n+1}$, $\mathbf{b} = (\mathbf{0}^l, \mathbf{1}^{n-l}, 0) \in \mathbb{R}^{n+1}$, $\mathbf{c} = (\frac{1}{l}\mathbf{1}^l, -\frac{1}{u}\mathbf{1}^{n-l}, 0) \in \mathbb{R}^{n+1}$, $\mathbf{A} = \mathbf{P}^\top \mathbf{K}^{-1}\mathbf{P}$, and $\gamma$ is determined by the following semi-definite programming:

$$\max_{\gamma, t, \alpha, \beta} \quad -\frac{1}{4}t + \sum_{i=1}^{n} \gamma_i - \epsilon(\alpha + \beta) \tag{9}$$

$$s.\,t. \quad \begin{pmatrix} \mathbf{A} - \mathcal{D}(\gamma \circ \mathbf{b}) & \gamma \circ \mathbf{a} - (\alpha - \beta)\mathbf{c}, \\ (\gamma \circ \mathbf{a} - (\alpha - \beta)\mathbf{c})^\top & t \end{pmatrix} \succeq 0$$

$$\alpha \geq 0, \; \beta \geq 0, \; \gamma_i \geq 0, \; i = 1, 2, \ldots, n.$$

**Proof Sketch**. We define the Lagrangian of the minimization problem (7) as follows:

$$\min_{\mathbf{w}} \ \max_{\gamma} \ \mathcal{F}(\mathbf{w}, \gamma) \ = \ \mathbf{w}^\top \mathbf{P}^\top \mathbf{K}^{-1} \mathbf{P} \mathbf{w} + \sum_{i=1}^{l} \gamma_i (1 - y_i^\ell w_i) + \sum_{i=l+1}^{n} \gamma_i (1 - w_i^2)$$
$$+ \alpha (\mathbf{c}^\top \mathbf{w} - \epsilon) + \beta(-\mathbf{c}^\top \mathbf{w} - \epsilon),$$

where $\gamma_i \geq 0$ for $i = 1, \dots, n$. It can be derived from the duality that $\min_{\mathbf{w}} \ \max_{\gamma} \ \mathcal{F}(\mathbf{w}, \gamma) = \max_{\gamma} \ \min_{\mathbf{w}} \ \mathcal{F}(\mathbf{w}, \gamma)$.

At the optimum, the derivatives of $\mathcal{F}$ with respect to the variable $\mathbf{w}$ are derived as below:

$$\frac{\partial \mathcal{F}}{\partial \mathbf{w}} = 2 \left[ \mathbf{A} - \mathcal{D}(\gamma \circ \mathbf{b}) \right] \mathbf{w} - \gamma \circ \mathbf{a} + (\alpha - \beta)\mathbf{c} = 0,$$

where $\mathbf{A} = \mathbf{P}^\top \mathbf{K}^{-1} \mathbf{P}$. The inverse of $\mathbf{A} - \mathcal{D}(\gamma \circ \mathbf{b})$ can be computed through adding a regularization parameter. Therefore, $\mathbf{w}$ is able to be calculated by:

$$\mathbf{w} = \frac{1}{2} \left[ \mathbf{A} - \mathcal{D}(\gamma \circ \mathbf{b}) \right]^{-1} (\gamma \circ \mathbf{a} - (\alpha - \beta)\mathbf{c}).$$

Thus, the dual form of the problem becomes:

$$\max_{\gamma} \quad \mathcal{L}(\gamma) = -\frac{1}{4}(\gamma \circ \mathbf{a} - (\alpha - \beta)\mathbf{c})^\top \left[ \mathbf{A} - \mathcal{D}(b \circ \gamma) \right]^{-1} (\gamma \circ \mathbf{a} - (\alpha - \beta)\mathbf{c}) + \sum_{i=1}^{n} \gamma_i - \epsilon(\alpha + \beta),$$

We import a variable $t$, so that

$$-\frac{1}{4}(\gamma \circ \mathbf{a} - (\alpha - \beta)\mathbf{c})^\top \left[ \mathbf{A} - \mathcal{D}(\mathbf{b} \circ \gamma) \right]^{-1} (\gamma \circ \mathbf{a} - (\alpha - \beta)\mathbf{c}) \geq -t.$$

According to the Schur Complement, we obtain a semi-definite programming cone, from which the optimization problem (9) can be formulated. ∎

**Remark I.** The problem in (9) is a convex optimization problem, more specifically, a semi-definite programming problem, and can be efficiently solved by the interior-point method [10] implemented in some optimization packages, such as SeDuMi [12]. Besides, our relaxation algorithm has $\mathcal{O}(n)$ parameters in the SDP cone and $\mathcal{O}(n)$ linear equality constraints, which involves a worst-case computational complexity of $\mathcal{O}(n^{4.5})$. However, in the previous relaxation algorithms [1, 14], there are approximately $\mathcal{O}(n^2)$ parameters in the SDP cone, which involve a worst-case computational complexity in the scale of $\mathcal{O}(n^{6.5})$. Therefore, our proposed convex relaxation algorithm is more efficient. In addition, as analyzed in Section 2, the approximation in [1, 14] drops the rank constraint of the matrix $\mathbf{y}^\top \mathbf{y}$, which does not lead to a tight approximation. On the other hand, our prediction function $f^*$ implements the conjugate of conjugate of the prediction function $f(\mathbf{x})$, which is the convex envelope of $f(\mathbf{x})$ [7]. Thus, our proposed convex approximation method provides a tighter approximation than the previous method.

**Remark II.** It is interesting to discuss the connection between the solution of the proposed algorithm and that of harmonic functions. We consider a special case of (8), where $\lambda = 0$ (which implies no bias term in the primal SVM), and there is no balance constraint. Then the solution of (9) can be expressed as follows:

$$\mathbf{z} = \frac{1}{2} \left[ \mathbf{K}^{-1} - \mathcal{D}(\gamma \circ (\mathbf{0}^l, \mathbf{1}^{n-l})) \right]^{-1} (\gamma \circ (\mathbf{y}^l, \mathbf{0}^{n-l})). \tag{10}$$

It can be further derived as follows:

$$\mathbf{z} = \left( \mathbf{I}_n - \sum_{i=l+1}^{n} \gamma_i \mathbf{K} \mathbf{I}_n^i \right)^{-1} \left( \sum_{i=1}^{l} \gamma_i y_i^\ell \mathbf{K}(\mathbf{x}_i, \cdot) \right), \tag{11}$$

where $\mathbf{I}_n^i$ is defined as an $n \times n$ matrix with all elements being zero except the $i$-th diagonal element which is 1, and $\mathbf{K}(\mathbf{x}_i, \cdot)$ is the $i$-th column of $\mathbf{K}$. Similar to the solution of the harmonic function, we first propagate the class labels from the labeled examples to the unlabeled one by term $\sum_{i=1}^{l} \gamma_i y_i^\ell \mathbf{K}(\mathbf{x}_i, \cdot)$, and then adjust the prediction labels by the factor $\left( \mathbf{I}_n - \sum_{i=l+1}^{n} \gamma_i \mathbf{K} \mathbf{I}_n^i \right)^{-1}$. The key difference in our solution is that (1) different weights (i.e., $\gamma_i$) are assigned to the labeled examples, and (2) the adjustment factor is different to that in the harmonic function [16].

### 3.2 Soft Margin TSVM

We extend TSVM to the case of soft margin by considering the following problem:

$$\min_{\nu,\mathbf{y},\delta,\lambda} \quad \frac{1}{2}(\mathbf{e} + \nu - \delta + \lambda\mathbf{y})^\top \mathcal{D}(\mathbf{y})\mathbf{K}^{-1}\mathcal{D}(\mathbf{y})(\mathbf{e} + \nu - \delta + \lambda\mathbf{y}) + C_\ell \sum_{i=1}^{l} \delta_i^2 + C_u \sum_{i=l+1}^{n} \delta_i^2$$

$$s.t. \quad \nu \geq 0, \; \delta \geq 0,$$
$$y_i = y_i^\ell, \; 1 \leq i \leq l,$$
$$y_i^2 = 1, \; l+1 \leq i \leq n,$$

where $\delta_i$ is related to the margin error. Note that we distinguish the labeled examples from the unlabeled examples by introducing different penalty constants for margin errors, $C_\ell$ for labeled examples and $C_u$ for unlabeled examples.

Similarly, we introduce the slack variable $\mathbf{z}$, and then derive the following dual problem:

$$\max_{\gamma,t,\alpha,\beta} \quad -\frac{1}{4}t + \sum_{i=1}^{n} \gamma_i - \epsilon(\alpha + \beta) \tag{12}$$

$$s.t. \quad \begin{pmatrix} \mathbf{A} - \mathcal{D}(\gamma \circ \mathbf{b}) & \gamma \circ \mathbf{a} - (\alpha - \beta)\mathbf{c} \\ (\gamma \circ \mathbf{a} - (\alpha - \beta)\mathbf{c})^\top & t \end{pmatrix} \succeq 0,$$
$$0 \leq \gamma_i \leq C_\ell, \; i = 1, 2, \ldots, l,$$
$$0 \leq \gamma_i \leq C_u, \; i = l+1, l+2, \ldots, n,$$
$$\alpha \geq 0, \; \beta \geq 0,$$

which is also a semi-definite programming problem and can be solved similarly.

## 4 Experiments

In this section, we report empirical study of the proposed method on several benchmark data sets.

### 4.1 Data Sets Description

To make evaluations comprehensive, we have collected four UCI data sets and three text data sets as our experimental testbeds. The UCI data sets include Iono, sonar, Banana, and Breast, which are widely used in data classification. The WinMac data set consists of the classes, mswindows and mac, of the Newsgroup20 data set. The IBM data set contains the classes, IBM and non-IBM, of the Newsgroup20 data set. The course data set is made of the course pages and non-course pages of the WebKb corpus. For each text data set, we randomly sample the data with the sample size of 60, 300 and 1000, respectively. Each resulted sample is noted by the suffix, "-s", "-m", or "-l" depending on whether the sample size is small, medium or large. Table 1 describes the information of these data sets, where $d$ represents the data dimensionality, $l$ means the number of labeled data points, and $n$ denotes the total number of examples.

Table 1: Data sets used in the experiments, where $d$ represents the data dimensionality, $l$ means the number of labeled data points, and $n$ denotes the total number of examples.

| Data set | $d$ | $l$ | $n$ | Data set | $d$ | $l$ | $n$ |
|----------|-----|-----|-----|----------|-----|-----|-----|
| Iono | 34 | 20 | 351 | WinMac-m | 7511 | 20 | 300 |
| Sonar | 60 | 20 | 208 | IBM-m | 11960 | 20 | 300 |
| Banana | 4 | 20 | 400 | Course-m | 1800 | 20 | 300 |
| Breast | 9 | 20 | 300 | WinMac-l | 7511 | 50 | 1000 |
| IBM-s | 11960 | 10 | 60 | IBM-l | 11960 | 50 | 1000 |
| Course-s | 1800 | 10 | 60 | Course-l | 1800 | 50 | 1000 |

### 4.2 Experimental Protocol

To evaluate the effectiveness of the proposed CTSVM method, we choose the conventional SVM as our baseline method. In our experiments, we also make comparisons with three state-of-the-art

methods: the SVM-light algorithm [8], the Gradient Decent TSVM ($\nabla$TSVM) algorithm [5], and the Concave Convex Procedure (CCCP) [6]. Since the SDP approximation TSVM [14] has very high time complexity $O(n^{6.5})$, which is difficult to process data sets with hundreds of examples. Thus, it is only evaluated on the smaller data sets, i.e., "IBM-s" and "Course-s".

The experiment setup is described as follows. For each data set, we conduct 10 trials. In each trial, the training set contains each class of data, and the remaining data are then used as the unlabeled (test) data. Moreover, the RBF kernel is used for "Iono", "Sonar" and "Banana", and the linear kernel is used for the other data sets. This is because the linear kernel performs better than the RBF kernel on these data sets. The kernel width of RBF kernel is chosen by 5-cross validation on the labeled data. The margin parameter $C_\ell$ is tuned by using the labeled data in all algorithms. Due to the small number of labeled examples, for CTSVM and CCCP, the margin parameter for unlabeled data, $C_u$, is set equal to $C_\ell$. Other parameters in these algorithms are set to the default values according to the relevant literatures.

### 4.3 Experimental Results

Table 2: The classification performance of Transductive SVMs on benchmark data sets.

| Data Set | SVM | SVM-light | $\nabla$TSVM | CCCP | CTSVM |
|---|---|---|---|---|---|
| Iono | 78.55$\pm$4.83 | 78.25$\pm$0.36 | 81.72$\pm$4.50 | **82.11**$\pm$3.83 | 80.09$\pm$2.63 |
| Sonar | 51.76$\pm$5.05 | 55.26$\pm$5.88 | **69.36**$\pm$4.69 | 56.01$\pm$6.70 | 67.39$\pm$6.26 |
| Banana | 58.45$\pm$7.15 | - | 71.54$\pm$7.28 | 79.33$\pm$4.22 | **79.51**$\pm$3.02 |
| Breast | 96.46$\pm$1.18 | 95.68$\pm$1.82 | 97.17$\pm$0.35 | 96.89$\pm$0.67 | **97.79**$\pm$0.23 |
| IBM-s | 52.75$\pm$15.01 | 67.60$\pm$9.29 | 65.80$\pm$6.56 | 65.62$\pm$14.83 | **75.25**$\pm$7.49 |
| Course-s | 63.52$\pm$5.82 | 76.82$\pm$4.78 | 75.80$\pm$12.87 | 74.20$\pm$11.50 | **79.75**$\pm$8.45 |
| WinMac-m | 57.64$\pm$9.58 | 79.42$\pm$4.60 | 81.03$\pm$8.23 | 84.28$\pm$8.84 | **84.82**$\pm$2.12 |
| IBM-m | 53.00$\pm$6.83 | 67.55$\pm$6.74 | 64.65$\pm$13.38 | 69.62$\pm$11.03 | **73.17**$\pm$0.89 |
| Course-m | 80.18$\pm$1.27 | **93.89**$\pm$1.49 | 90.35$\pm$3.59 | 88.78$\pm$2.87 | 92.92$\pm$2.28 |
| WinMac-l | 60.86$\pm$10.10 | 89.81$\pm$2.10 | 90.19$\pm$2.65 | 91.00$\pm$2.42 | **91.25**$\pm$2.67 |
| IBM-l | 61.82$\pm$7.26 | **75.40**$\pm$2.26 | 73.11$\pm$1.99 | 74.80$\pm$1.87 | 73.42$\pm$3.23 |
| Course-l | 83.56$\pm$3.10 | 92.35$\pm$3.02 | 93.58$\pm$2.68 | 91.32$\pm$4.08 | **94.62**$\pm$0.97 |

Table 2 summarizes the classification accuracy and the standard deviations of the proposed algorithm, the baseline method and the state-of-the-art methods. It can be observed that our proposed algorithm performs significantly better than the standard SVM across all the data sets. Moreover, on the small-size data sets, i.e., "IBM-s" and "Course-s", the results of the SDP-relaxation method are 68.57$\pm$22.73 and 64.03$\pm$7.65, which are worse than the proposed CTSVM method. In addition, the proposed CTSVM algorithm performs much better than other TSVM methods over "WinMac-m" and "Course-l". As shown in Table 2, the SVM-light algorithm achieves the best results on "Course-m" and "IBM-l", however, it fails to converge on "Banana". On the remaining data sets, comparable results have been obtained for our proposed algorithm. From above, the empirical evaluations indicate that our proposed CTSVM method achieves promising classification results comparing with the state-of-the-art methods.

## 5   Conclusion and Future Work

This paper presents a novel method for Transductive SVM by relaxing the unknown labels to the continuous variables. In contrast to the previous relaxation method which involves $\mathcal{O}(n^2)$ free parameters in the semi-definite matrix, our method takes the advantages of reducing the number of free parameters to $\mathcal{O}(n)$, and can solve the optimization problem more efficiently. In addition, the proposed approach provides a tighter convex relaxation for the optimization problem in TSVM. Empirical studies on benchmark data sets demonstrate that the proposed method is more efficient than the previous semi-definite relaxation method and achieves promising classification results comparing to the state-of-the-art methods.

As the current model is only designed for a binary-classification, we plan to develop a multi-class Transductive SVM model in the future. Moreover, it is desirable to extend the current model to classify the new incoming data.

## Acknowledgments

The work described in this paper is supported by a CUHK Internal Grant (No. 2050346) and a grant from the Research Grants Council of the Hong Kong Special Administrative Region, China (Project No. CUHK4150/07E).

## References

[1] T. D. Bie and N. Cristianini. Convex methods for transduction. In S. Thrun, L. Saul, and B. Schölkopf, editors, *Advances in Neural Information Processing Systems 16*. MIT Press, Cambridge, MA, 2004.

[2] O. Chapelle, M. Chi, and A. Zien. A continuation method for semi-supervised SVMs. In *ICML '06: Proceedings of the 23rd international conference on Machine learning*, pages 185–192, New York, NY, USA, 2006. ACM Press.

[3] O. Chapelle, B. Schölkopf, and A. Zien. *Semi-Supervised Learning*. MIT Press, Cambridge, MA, 2006.

[4] O. Chapelle, V. Sindhwani, and S. Keerthi. Branch and bound for semi-supervised support vector machines. In B. Schölkopf, J. Platt, and T. Hoffman, editors, *Advances in Neural Information Processing Systems 19*. MIT Press, Cambridge, MA, 2007.

[5] O. Chapelle and A. Zien. Semi-supervised classification by low density separation. In *Proceedings of the Tenth International Workshop on Artificial Intelligence and Statistics*, pages 57–64, 2005.

[6] R. Collobert, F. Sinz, J. Weston, and L. Bottou. Large scale transductive SVMs. *Journal of Machine Learning Reseaerch*, 7:1687–1712, 2006.

[7] J.-B. Hiriart-Urruty and C. Lemarechal. *Convex analysis and minimization algorithms II: advanced theory and bundle methods. (2nd part edition).* Springer-Verlag, New York, 1993.

[8] T. Joachims. Transductive inference for text classification using support vector machines. In *ICML '99: Proceedings of the Sixteenth International Conference on Machine Learning*, pages 200–209, San Francisco, CA, USA, 1999. Morgan Kaufmann Publishers Inc.

[9] G. R. G. Lanckriet, N. Cristianini, P. Bartlett, L. E. Ghaoui, and M. I. Jordan. Learning the kernel matrix with semidefinite programming. *Journal of Machine Learning Research*, 5:27–72, 2004.

[10] Y. Nesterov and A. Nemirovsky. *Interior point polynomial methods in convex programming: Theory and applications*. Studies in Applied Mathematics. Philadelphia, 1994.

[11] V. Sindhwani, S. S. Keerthi, and O. Chapelle. Deterministic annealing for semi-supervised kernel machines. In *ICML '06: Proceedings of the 23rd international conference on Machine learning*, pages 841–848, New York, NY, USA, 2006. ACM Press.

[12] J. F. Sturm. Using SeDuMi 1.02, a MATLAB toolbox for optimization over symmetric cones. *Optimization Methods and Software*, 11:625–653, 1999.

[13] H. Valizadegan and R. Jin. Generalized maximum margin clustering and unsupervised kernel learning. In B. Schölkopf, J. Platt, and T. Hoffman, editors, *Advances in Neural Information Processing Systems 19*. MIT Press, Cambridge, MA, 2007.

[14] L. Xu and D. Schuurmans. Unsupervised and semi-supervised multi-class support vector machines. In *AAAI*, pages 904–910, 2005.

[15] X. Zhu. Semi-supervised learning literature survey. Technical report, Computer Sciences, University of Wisconsin-Madison, 2005.

[16] X. Zhu, Z. Ghahramani, and J. D. Lafferty. Semi-supervised learning using gaussian fields and harmonic functions. In *Proceedings of Twentith International Conference on Machine Learning (ICML-2003)*, pages 912–919, 2003.
